# Active Instance Sampling via Matrix Partition

**Yuhong Guo**
Department of Computer & Information Sciences
Temple University
Philadelphia, PA 19122
`yuhong@temple.edu`

## Abstract

Recently, batch-mode active learning has attracted a lot of attention. In this paper, we propose a novel batch-mode active learning approach that selects a batch of queries in each iteration by maximizing a natural mutual information criterion between the labeled and unlabeled instances. By employing a Gaussian process framework, this mutual information based instance selection problem can be formulated as a matrix partition problem. Although matrix partition is an NP-hard combinatorial optimization problem, we show that a good local solution can be obtained by exploiting an effective local optimization technique on a relaxed continuous optimization problem. The proposed active learning approach is independent of employed classification models. Our empirical studies show this approach can achieve comparable or superior performance to discriminative batch-mode active learning methods.

## 1 Introduction

Active learning is well-motivated in many supervised learning scenarios where unlabeled instances are abundant and easy to retrieve but labels are difficult, time-consuming, or expensive to obtain. For example, it is easy to gather large amounts of unlabeled documents or images from the Internet, whereas labeling them requires manual effort from experienced human annotators. Randomly selecting unlabeled instances for labeling is inefficient in many situations, since non-informative or redundant instances might be selected. Aiming to reduce labeling effort, active learning (i.e., selective sampling) methods have been adopted to control the labeling process in many areas of machine learning. Given a large pool of unlabeled instances, active learning provides a way to iteratively select the most informative unlabeled instances—the queries—from the pool to label.

Many researchers have addressed the active learning problem in various ways [13]. Most have focused on selecting a single most informative unlabeled instance to query each time. The ultimate goal for most such approaches is to select instances that could lead to a classifier with low generalization error. Towards this, a few variants of a mutual information criterion have been employed in the literature to guide the active instance sampling process. The approaches in [4][10] select the instance to maximize the increase of mutual information and the mutual information, respectively, between the selected set of instances and the remainder based on Gaussian process models. The approach proposed in [5] seeks the instance whose optimistic label provides maximum mutual information about the labels of the remaining unlabeled instances. The mutual information measure used is discriminative, computed using their trained classifier at that point. This approach implicitly exploits the clustering information contained in the unlabeled data in an optimistic way.

The single instance selection active learning methods require tedious retraining with each single instance being labeled. When the learning task is sufficiently complex, the retraining process between queries can become very slow. This may make highly interactive learning inefficient or impractical. Furthermore, if a parallel labeling system is available, e.g., multiple annotators working on

different labeling workstations at the same time on a network, a single instance selection system can make wasteful use of the resource. Thus, a batch-mode active learning strategy that selects multiple instances each time is more appropriate under these circumstances. The challenge in batch-mode active learning is how to properly assemble the optimal query batch. Simply using a single instance selection strategy to select a batch of queries in each iteration does not work well, since it fails to take the information overlap between the multiple instances into account. Principles for batch mode active learning need to be developed to address the multi-instance selection specifically. Several sophisticated batch-mode active learning methods have been proposed for classification. Most of these approaches use greedy heuristics to ensure the overall informativeness of the batch by taking both the individual informativeness and the diversity of the selected instances into account. Schohn and Cohn [12] select instances according to their proximity to the dividing hyperplane for a linear SVM. Brinker [2] considers an approach for SVMs that explicitly takes the diversity of the selected instances into account, in addition to individual informativeness. Xu et al. [14] propose a representative sampling approach for SVM active learning, which also incorporates a diversity measure. Specifically, they query cluster centroids for instances that lie close to the decision boundary. Hoi et al. [7, 8] extend the Fisher information framework to the batch-mode setting for binary logistic regression. Hoi et al. [9] propose a novel batch-mode active learning scheme on SVMs that exploits semi-supervised kernel learning. In particular, a kernel function is first learned from a mixture of labeled and unlabeled examples, and then is used to effectively identify the informative and diverse instances via a min-max framework. Instead of using heuristic measures, Guo and Schuurmans [6] treat batch construction for logistic regression as a discriminative optimization problem, and attempt to construct the most informative batch directly. Overall, these batch-mode active learning approaches all make batch selection decisions directly based on the classifiers employed.

In this paper, we propose a novel batch-mode active learning approach that makes query selection decisions independent of the classification model employed. The idea is to select a batch of queries in each iteration by maximizing a general *mutual information* measure between the labeled instances and the unlabeled instances. By employing a Gaussian process framework, this mutual information maximization problem can be further formulated as a matrix partition problem. Although the matrix partition problem is an NP-hard combinatorial optimization, it can first be relaxed into a continuous optimization problem and then a good local solution can be obtained by exploiting an effective local optimization. The local optimization method we use is developed by combining a local linearization of the objective function based on its first-order Taylor series expansion, and a straightforward backtracking line search. Unlike most active learning methods studied in the literature, our query selection method does not require knowledge of the employed classifier. Our empirical studies show that the proposed batch-mode active learning approach can achieve superior or comparable performance to discriminative batch-mode active learning methods that have been optimized on specific classifiers.

The remainder of the paper is organized as follows. Section 2 provides preliminaries on Gaussian processes. Section 3 introduces the proposed matrix partition approach for batch-mode active learning. Empirical studies are presented in Section 4, and Section 5 concludes this work.

## 2  Gaussian Processes

A Gaussian process is a generalization of the Gaussian probability distribution. Although Gaussian processes have a long history in statistics, their potential has only become widely appreciated in the machine learning community during the past decade [11]. In this section, we provide an overview of Gaussian processes and some of their important properties which we will exploit later to construct our active learning approach.

### 2.1  Multivariate Gaussian Distribution

The Gaussian, also known as the normal distribution, is a widely used model for the distribution of continuous variables. In the case of multiple random variables, the joint multivariate Gaussian distribution for a $d \times 1$ vector $\mathbf{x}$ is given in the form

$$P(\mathbf{x}) = \frac{1}{(2\pi)^{d/2}|\Sigma|^{1/2}} \exp\left(-\frac{1}{2}(\mathbf{x} - \boldsymbol{\mu})^{\top}\Sigma^{-1}(\mathbf{x} - \boldsymbol{\mu})\right)$$

where $\boldsymbol{\mu}$ is a $d$-dimensional mean vector, $\Sigma$ is a $d \times d$ covariance matrix, and $|\Sigma|$ denotes the determinant of $\Sigma$. When $d = 1$, we obtain the standard one-variable Gaussian distribution.

## 2.2  Gaussian Processes

A Gaussian process is a generalization of a multivariate Gaussian distribution over a finite vector space to a function space of infinite dimension. Given a set of instances $\mathbf{X} = [\mathbf{x}_1^\top; \mathbf{x}_2^\top; \cdots; \mathbf{x}_t^\top]$, a data modeling function $f(\cdot)$ can be viewed as a single sample from a Gaussian distribution with a mean function $\mu(\cdot)$, and a covariance function $C(\cdot, \cdot)$. In particular, $\mu(\mathbf{x}_i)$ denotes the mean of the function variable $f(\mathbf{x}_i)$ at point $\mathbf{x}_i$, and $C(\mathbf{x}_i, \mathbf{x}_j)$ expresses the expected covariance between functions $f$ at point $\mathbf{x}_i$ and $\mathbf{x}_j$. A Gaussian process is defined as a Gaussian distribution on a space of functions $f$ which can be written in the form

$$P(f(\mathbf{x})) = \frac{1}{Z} \exp\left(-\frac{1}{2}(f(\mathbf{x}) - \mu(\mathbf{x}))^\top \Sigma^{-1}(f(\mathbf{x}) - \mu(\mathbf{x}))\right)$$

where $\mu(\mathbf{x})$ is the mean function, $\Sigma$ is defined using the covariance function $C$, and $Z$ denotes the normalization factor. One typical choice for the covariance function $C$ is a symmetric positive-definite kernel function $\mathcal{K}$, e.g. a Gaussian kernel

$$\mathcal{K}(\mathbf{x}_i, \mathbf{x}_j) = \exp\left(-\frac{(\|\mathbf{x}_i - \mathbf{x}_j\|^2)}{\tau^2}\right) \tag{1}$$

One important property of Gaussian processes is that for every finite set (or subset) of instances $\mathbf{X}_Q$ with indices $Q$, the joint distribution over the corresponding random function variables $\mathbf{f}_Q = \mathbf{f}(\mathbf{X}_Q)$ is a multivariate Gaussian distribution with a mean vector $\mu_Q = \mu(\mathbf{X}_Q)$ and a covariance matrix $\Sigma_{QQ}$, where each entry $\Sigma_{i,j}$ is defined using the covariance kernel function $\mathcal{K}(\mathbf{x}_i, \mathbf{x}_j)$

$$P(f_Q) = \frac{1}{Z} \exp\left(-\frac{1}{2}(f_Q - \mu_Q)^\top \Sigma_{QQ}^{-1}(f_Q - \mu_Q)\right) \tag{2}$$

Here $Z = (2\pi)^{q/2}|\Sigma_{QQ}|^{1/2}$, and $q$ is the size of set $Q$. We can assume the the mean function $\mu(\cdot) = 0$. Nevertheless, it is irrelevant in this paper.

# 3  Batch-mode Active Learning via Matrix Partition

Given a small set of labeled instances $\{(\mathbf{x}_i, y_i)\}_{i \in L}$ and a large set of unlabeled instances $\{\mathbf{x}_j\}_{j \in U}$, our task is to iteratively select the most informative set of $b$ instances from $U$ and add them into the labeled set $L$ after querying their labels from a labeling system. In this section, we propose to conduct instance selective sampling using a maximum mutual information strategy which can then be formulated into a matrix partition problem.

## 3.1  Maximum Mutual Information Instance Selection

Since the ultimate goal of active learning is to achieve a classifier with good generalization performance on unseen test data, it makes sense to select instances that can produce a labeled set that is most informative about the unseen test instances. Apparently it is not possible to access the *unseen* test data. Nevertheless, in active learning setting, we have a large number of unlabeled instances available that come from the same distribution as the future test instances. Thus we can select instances that lead to a labeled set which is most informative about the large set of unlabeled instances instead. We propose to use a mutual information criterion to measure the informativeness of the labeled set $L$ over the unlabeled set $U$

$$I(\mathbf{X}_L, \mathbf{X}_U) = H(\mathbf{X}_L) + H(\mathbf{X}_U) - H(\mathbf{X}_L, \mathbf{X}_U) \tag{3}$$

where $\mathbf{X}_L$ and $\mathbf{X}_U$ denotes the labeled set of instances and the unlabeled set of instances respectively, $H(\cdot)$ denotes the entropy term.

Both the mutual information measure and the entropy measure are defined on probability distributions [3]. We thus employ a Gaussian process framework (introduced in the previous section) to

model the joint probability distribution over all the instances. We first associate each instance $\mathbf{x}_i$ with a random variable $f_i$. Then the joint distribution over a finite number of instances $\mathbf{X}_Q$ can be represented using the joint multivariate Gaussian distribution over variables $f_Q$, which is given in (2). Thus the entropy term $H(\mathbf{X}_Q) = H(f_Q)$ can be computed using a closed-form solution

$$H(f_Q) = \frac{1}{2} \ln \left( (2\pi e)^m |\Sigma_{QQ}| \right) \tag{4}$$

where $m$ is the number of variables, i.e., the size of $Q$; $\Sigma_{QQ}$ is the covariance matrix computed over $\mathbf{X}_Q$ using a kernel function $\mathcal{K}$ given in (1). Within this Gaussian process framework, the mutual information criterion in (3) can be rewritten as

$$
\begin{aligned}
I(\mathbf{X}_L, \mathbf{X}_U) &= H(f_L) + H(f_U) - H(f_L, f_U) \\
&= \frac{1}{2} \ln \left( (2\pi e)^l |\Sigma_{LL}| \right) + \frac{1}{2} \ln \left( (2\pi e)^u |\Sigma_{UU}| \right) - \frac{1}{2} \ln \left( (2\pi e)^t |\Sigma_{VV}| \right)
\end{aligned}
\tag{5}
$$

where $V$ is the union of $L$ and $U$; $l, u, t$ denote the sizes of $L, U, V$ respectively such that $l + u = t$. Note that for a given data set, the overall number of instances does not change during the active learning process. We simply move $b$ instances from the unlabeled set $U$ into the labeled set $L$ in each iteration. Thus the set $V$ and the entropy term $H(f_L, f_U)$ are irrelevant to the instance selection. Based on this observation, our maximum mutual information instance selection strategy can be formulated as

$$Q^* = \underset{|Q|=b, Q \subseteq U}{\arg\max} I(\mathbf{X}_{L \cup Q}, \mathbf{X}_{U \setminus Q}) = \underset{|Q|=b, Q \subseteq U}{\arg\max} \ln |\Sigma_{L'L'}| + \ln |\Sigma_{U'U'}| \tag{6}$$

where $L' = L \cup Q$ and $U' = U \setminus Q$. This also suggests the mutual information criterion depends only on the covariance matrices computed using the kernel functions over the instances. Our maximum mutual information strategy attempts to select the batch of $b$ instances from the unlabeled set $U$ to label, to maximize the log determinants of the covariance matrices over the produced sets $L'$ and $U'$.

## 3.2   Matrix Partition

Let $\Sigma$ be the covariance matrix over all the instances indexed by $V = L \cup U = L' \cup U'$. Then the covariance matrices $\Sigma_{LL}$, $\Sigma_{UU}$, $\Sigma_{L'L'}$ and $\Sigma_{U'U'}$ are all submatrices of $\Sigma$. Without losing any generality, we assume the instances are arranged in the order of $[U, L]$, such that

$$\Sigma = \begin{bmatrix} \Sigma_{UU} & \Sigma_{UL} \\ \Sigma_{LU} & \Sigma_{LL} \end{bmatrix} \tag{7}$$

The instance selection problem formulated in (6) selects a subset of $b$ instances indexed by $Q$ from $U$ and moves them into the labeled set $L$. This problem is actually equivalent to *partitioning matrix* $\Sigma$ into submatrices $\Sigma_{L'L'}$, $\Sigma_{U'U'}$, $\Sigma_{L'U'}$ and $\Sigma_{U'L'}$ by reordering the instances in $U$. Since $L$ is fixed, the actual matrix partition is conduct on covariance matrix $\Sigma_{UU}$. Now we define a permutation matrix $M \in \{0,1\}^{u \times u}$ such that

$$M\mathbf{1} = \mathbf{1}, \quad M^\top \mathbf{1} = \mathbf{1}$$

where $\mathbf{1}$ denotes a vector of all $1$ entries. We let $M_{\bar{b}}$ denote the first $u - b$ rows of $M$, and $M_b$ denote the last $b$ rows of $M$, such that

$$M_{\bar{b}} \Sigma_{UU} M_{\bar{b}}^\top = \Sigma_{U'U'}, \quad M_b \Sigma_{UU} M_b^\top = \Sigma_{QQ} \tag{8}$$

Obviously $M_b$ selects $b$ instances from $U$ to form $Q$. Let

$$T = \begin{bmatrix} M_{\bar{b}} & O_{(u-b) \times l} \end{bmatrix}, \quad B = \begin{bmatrix} M_b & O_{b \times l} \\ O_{l \times u} & I_l \end{bmatrix} \tag{9}$$

where $O_{m \times n}$ denotes a $m \times n$ matrix with all $0$ entries, and $I_l$ denotes a $l \times l$ identity matrix. According to (8) we then have

$$\Sigma_{U'U'} = T\Sigma T^\top, \quad \Sigma_{L'L'} = B\Sigma B^\top \tag{10}$$

Finally, the maximum mutual information problem given in (6) can be equivalently formulated into the following matrix partition problem

$$\max_{M} \quad \ln |B\Sigma B^\top| + \ln |T\Sigma T^\top| \tag{11}$$

$$\text{s.t.} \quad M \in \{0,1\}^{u \times u}, \; M\mathbf{1} = \mathbf{1}, \; M^\top \mathbf{1} = \mathbf{1}$$

After solving this problem to obtain an optimal $M^*$, the instance selection can be determined from the last $b$ rows of $M^*$, i.e., $M_b^*$.

However, the optimization problem (11) is an NP-hard combinatorial optimization problem over an integer matrix $M$. To facilitate a convenient optimization procedure, we relax the integer optimization problem (11) into the following upper bound optimization problem

$$\max_{M} \quad \ln|B\Sigma B^\top| + \ln|T\Sigma T^\top| \tag{12}$$

$$\text{s.t.} \quad 0 \le M \le 1, \ M\mathbf{1} = \mathbf{1}, \ M^\top\mathbf{1} = \mathbf{1} \tag{13}$$

Note a determinant is a log concave function on positive definite matrices [1]. Thus $\ln|X|$ is concave in $X$. However, the quadratic matrix function $X = B\Sigma B^\top$ is matrix convex given the matrix $\Sigma$ is positive definite. Thus the composition function $\ln|B\Sigma B^\top|$ is neither convex nor concave, but differentiable. In general, this type of problems are difficult global optimization problems. We develop an efficient local optimization technique to solve for a reasonable local solution instead.

### 3.3 First-order Local Optimization

The target optimization (12) is an optimization problem over a $u \times u$ matrix $M$, subject to the linear inequality and equality constraints (13). Here $u$ is the number of unlabeled instances, and we typically assume it is a large number. Therefore a second-order optimization approach will be space demanding. We develop a first-order local maximization algorithm to conduct optimization, which combines a gradient direction finding method with a straightforward backtracking line search technique. This local optimization algorithm produced promising results in our experiments.

The algorithm is an iterative procedure, starting from an initial matrix $M^{(0)}$. Let $M^{(k)}$ denote the optimization variable values returned from the the $k$th iteration. At the $(k+1)$th iteration, we approximate the objective function in (12) using its first-order Taylor series expansion at point $M^{(k)}$

$$
\begin{aligned}
g(M) &= \ln|B\Sigma B^\top| + \ln|T\Sigma T^\top| \\
&\approx \ln|B^{(k)}\Sigma B^{(k)\top}| + \ln|T^{(k)}\Sigma T^{(k)\top}| + \text{Tr}\left(G(M^{(k)})^\top(M - M^{(k)})\right)
\end{aligned} \tag{14}
$$

Where $B^{(k)}$ and $T^{(k)}$ denote the corresponding $B$ and $T$ matrices with their $M$ submatrices fixed to values given by $M^{(k)}$; Tr denotes the trace operator; $G(M^{(k)})$ denotes the gradient matrix value at point $M^{(k)}$. The gradient of the objective function $g(M)$ can be calculated using the matrix calculus, which gives the following results

$$G(M_{\bar{b}}) = \frac{dg(M)}{dM_{\bar{b}}} = 2\big[(T\Sigma T^\top)^{-1}T\Sigma\big]_{1:(u-b),1:u} \tag{15}$$

$$G(M_b) = \frac{dg(M)}{dM_b} = 2\big[(B\Sigma B^\top)^{-1}B\Sigma\big]_{1:b,1:u} \tag{16}$$

$$G(M) = \big[G(M_{\bar{b}})^\top, G(M_b)^\top\big]^\top \tag{17}$$

Note here we use notations in the matlab format where $[X]_{i:j,m:n}$ denotes the $(j-i+1)\times(n-m+1)$ submatrix of $X$ formed by entries between the $i$th to the $j$th rows and the $m$th to the $n$th columns.

Given the gradient at point $M^{(k)}$, we maximize the local linearization (14) to seek a gradient direction regarding the constraints. This leads to a convex linear optimization

$$\widetilde{M} = \arg\max_{M} \ \text{Tr}\left(G(M^{(k)})^\top M\right) \tag{18}$$

$$\text{s.t.} \quad 0 \le M \le 1, \ M\mathbf{1} = \mathbf{1}, \ M^\top\mathbf{1} = \mathbf{1}$$

The gradient direction for the $(k+1)$th iteration can be determined as

$$D = \widetilde{M} - M^{(k)}. \tag{19}$$

We then employ a backtracking line search to seek the optimal value $M^{(k+1)}$ to improve the original objective function $g(M)$ with $g(M^{(k+1)}) > g(M^{(k)})$. The line search procedure,

---

**Algorithm 1** Matrix Partition

---

**Input:** $l$: the number of labeled instances; $u$ the number of unlabeled instances;
   $\Sigma$: covariance matrix given in form of (7); $b$: batch size;
   $M^{(0)}$; $\epsilon < 1e - 8$.
**Output:** $M^*$
**Initialize** $k = 0$, $NoChange = false$.
**repeat**
   Set $T$ and $B$ according to equations (9) using the current $M^{(k)}$.
   Compute gradient $G(M^{(k)})$ at point $M^{(k)}$ according to equations (15), (16) and (17).
   Solve the local linear optimization (18) for the given gradient to get $\widetilde{M}$.
   Compute the gradient ascend direction $D$ using the equation (19).
   Compute $M^{(k+1)} = linesearch(D, M^{(k)})$.
   **if** $\|M^{(k+1)} - M^{(k)}\|^2 < \epsilon$ **then** NoChange=true. **end if**
   k = k+1.
**until** $NoChange$ is true or maximum iteration number is reached.
$M^* = M^{(k)}$.

---

---

**Algorithm 2** Heuristic Greedy Rounding Procedure

---

**Input:** b, $M \in (0,1)^{b \times u}$ for $b < u$.
**Output:** $\widehat{M}, Q$.
**Initialize** Let $Q = \emptyset$, set $\widehat{M}$ as a $b \times u$ matrix with all 0 entries.
**for** $k = 1$ **to** $b$ **do**
   Identify the largest value $v = max(M(:))$.
   Identify the indices $(i, j)$ of $v$ in $M$.
   Set $Q = Q \cup \{j\}$, $\widehat{M}(i,j) = 1$, $M(i,:) = -\text{Inf}$, $M(:,j) = -\text{Inf}$.
**end for**

---

$linesearch(D, M^{(k)})$, seeks an optimal step size, $0 \le s < 1$, to update the $M^{(k)}$ in the ascending direction $D$ given in (19), i.e. $M^{(k+1)} = M^{(k)} + sD$, guaranteeing the returned $M^{(k+1)}$ satisfies the linear constraints in (13), and leads to an objective value no worse than before.

The overall algorithm for optimizing the matrix partition problem (12) is given in Algorithm 1. In our implementation, the constrained linear optimization (18) can be efficiently solved using an optimization software package CPLEX. When the number of unlabeled instances, $u$, is large, computing the log-determinant of the $(u-b) \times (u-b)$ matrix, $T\Sigma T^\top$, is likely to run into overflow or underflow. Instead of computing the log-determinant directly, we choose to compute it in an alternative efficient way. The key idea is based on the mathematical fact that the determinant of a triangular matrix equals the product of its diagonal elements. Hence, the matrix's log-determinant is equal to the sum of their logarithm values. By keeping all computations in log-scale, the problem of underflow/overflow caused by product of many numbers can be effectively circumvented. For positive definite matrices, such as the matrices we have, one can use Cholesky factorization to first produce a triangular matrix and then compute the log-determinant of the original matrix using the logarithms of the diagonal values of the triangular matrix. The computation of log-determinants or matrix inverse in our algorithm are all conducted on matrices assumed to be positive definite. However, in order to increase the robustness of the algorithm and avoid numerical problems, we can add an additional $\delta I$ term to the matrices to guarantee the positive definite property. Here $\delta$ is a very small value and $I$ is an identity matrix.

By solving the matrix partition problem in (12) using Algorithm 1, an optimal matrix $M^*$ is returned. However, this $M^*$ contains continuous values. In order to determine which set of $b$ instances to select, we need to round $M^*$ to a $\{0,1\}$-valued $\widehat{M^*}$, while maintaining the permutation constraints $\widehat{M^*}\mathbf{1} = \mathbf{1}$ and $\widehat{M^*}^\top\mathbf{1} = \mathbf{1}$. We use a simple heuristic greedy procedure to conduct the rounding. In this procedure, we focused on rounding the last $b$ rows, $M_b^*$, since they are the ones used to pick $b$ instances for labeling. The procedure is described in Algorithm 2, which returns the indices of the selected $b$ instances as well.

## 4 Experiments

To investigate the empirical performance of the proposed batch-mode active learning algorithm, we conducted two sets of experiments on a few UCI datasets and the 20 newsgroups dataset. Note the proposed active learning method is in general independent of the specific classification model employed. For the experiments in this section, we used logistic regression as its classification model to evaluate the informativeness of the selected labeled instances. We compared the proposed approach, denoted as *Matrix*, with three discriminative batch-mode active learning methods proposed in the literature: *svmD*, an approach that incorporates diversity in active learning with SVMs [2]; *Fisher*, an approach that uses Fisher information matrix based on logistic regression classifiers for instance selection [8]; *Discriminative*, a discriminative optimization approach based on logistic regression classifiers [6]. We have also compared our approach to one transductive experimental design method which is formulated from regression problems and whose instance selection process is independent of evaluation classification models [15]. We used the sequential design code downloaded from the authors' webpage and denote this method as *Design*.

First, we conducted experiments on seven UCI datasets. We consider a hard case of active learning, where we start active learning from only a few labeled instances. In each experiment, we start with two randomly selected labeled instances, one in each class. We then randomly select 2/3 of the remaining instances as the unlabeled set, using all the other instances for testing. All the algorithms start with the same initial labeled set, unlabeled set and testing set. For a fixed batch size $b$, each algorithm repeatedly select $b$ instances to label each time and evaluate the produced classifier on testing data after each new labeling, with maximum 110 instances to select in total. The experiments were repeated 20 times. In Table 1, we report the experimental results with $b = 10$, comparing the proposed *Matrix* algorithm with each of the three batch-mode alternatives. With $b = 10$, there are totally 11 evaluation points, with 20 results on each of them. We therefore run a 2-sided paired t-test at each evaluation point to compare the performance of each pair of algorithms. The "win%" denotes the percentage of evaluation points where the *Matrix* algorithm outperforms the specified algorithm using a 2-sided paired t-test at the level of p<0.05; the "lose%" denotes the percentage of evaluation points where the specified algorithm outperforms the *Matrix* algorithm. The "overall" nevertheless show the comparison results using a single 2-sided paired t-test on all 220 results. These results show that the proposed active learning method, *Matrix*, overperformed *svmD*, *Fisher* and *Design* on most data sets, except an overall *lose* to *svmD* on pima, a *tie* with *Fisher* and *Design* on hepatitis, and a *tie* with *Design* on flare. *Matrix* is mostly tied with *Discriminative* on all data sets, with a slight pointwise *win* on crx and a slight overall *lose* on german. Although *Matrix* and *Discriminative* demonstrated similar performance, the proposed *Matrix* is more efficient regarding running time on relatively big data sets. The comparison in running times over 20 repeats are reported in Table 2.

Table 1: Comparison of the active learning algorithms on UCI data with batch size = 10. These results are based on 2-sided paired t-test at the level of p< 0.05.

| Data set | Matrix *vs* svmD | | | Matrix *vs* Fisher | | | Matrix *vs* Discriminative | | | Matrix *vs* Design | | |
|---|---|---|---|---|---|---|---|---|---|---|---|---|
| | win% | lose% | overall | win% | lose% | overall | win% | lose% | overall | win% | lose% | overall |
| cleve | 63.6 | 0 | win | 45.5 | 0 | win | 0 | 0 | tie | 90.9 | 0 | win |
| crx | 27.3 | 0 | win | 9.1 | 0 | win | 9.1 | 0 | tie | 90.9 | 0 | win |
| flare | 54.5 | 0 | win | 100.0 | 0 | win | 0 | 0 | tie | 36.4 | 9.1 | tie |
| german | 81.8 | 0 | win | 9.1 | 0 | win | 0 | 0 | lose | 72.7 | 0 | win |
| heart | 63.6 | 0 | win | 36.4 | 0 | win | 0 | 0 | tie | 100.0 | 0 | win |
| hepatitis | 100.0 | 0 | win | 33.3 | 0 | tie | 0 | 0 | tie | 0 | 0 | tie |
| pima | 0 | 0 | lose | 100.0 | 0 | win | 0 | 0 | tie | 81.8 | 0 | win |

Table 2: Average running time (in minutes)

| Method | cleve | crx | flare | german | heart | hepatitis | pima |
|---|---|---|---|---|---|---|---|
| Matrix | 8.37 | 6.14 | 9.53 | 22.08 | 5.68 | 0.12 | 60.11 |
| Discriminative | 3.33 | 61.44 | 220.12 | 285.65 | 2.40 | 0.08 | 68.27 |

Table 3: Comparison of the active learning algorithms on Newsgroup data with batch size = 20. These results are based on 2-sided paired t-test at the level of p< 0.05.

| Data set | Matrix $vs$ svmD | | | Matrix $vs$ Fisher | | | Matrix $vs$ Random | | | Matrix $vs$ Design | | |
|---|---|---|---|---|---|---|---|---|---|---|---|---|
| | win% | lose% | overall | win% | lose% | overall | win% | lose% | overall | win% | lose% | overall |
| Autos | 86.7 | 0 | win | 20.0 | 6.6 | tie | 73.3 | 6.6 | win | 80.0 | 6.7 | win |
| Hardware | 100.0 | 0 | win | 0 | 0 | tie | 13.3 | 0 | win | 86.7 | 0 | win |
| Sport | 86.7 | 6.6 | win | 20.0 | 13.3 | tie | 46.7 | 0 | win | 80.0 | 6.7 | win |

Next we conducted experiments on 20 newsgroups dataset for document categorization. We build three binary classification tasks: (1) Autos: rec.autos (987 documents) vs. rec.motorcycles (993 documents); (2) Hardware: comp.sys.ibm.pc.hardware (979 documents) vs. comp.sys.mac.hardware (958 documents); (3) Sport: rec.sport.baseball (991 documents) vs. rec.sport.hockey (997 documents). Each document is first minimally processed into a "tf.idf" vector. We then select the top 400 features to use according to their total "tf.idf" frequencies in all the documents for the considered task. In each experiment, we start with four randomly selected labeled instances, two in each class. We then randomly select 1000 instances (500 from each class) from the remaining ones as the unlabeled set, using all the other instances for testing. All the algorithms start with the same initial labeled set, unlabeled set and testing set. For a fixed batch size $b$, each algorithm repeatedly select $b$ instances to label each time with maximum 300 instances to select in total. In this section, we report the experimental results with $b = 20$ averaged over 20 times repetitions. There are $300/20 = 15$ evaluation points in this case.

Note the unlabeled sets used for this set of experiments are much larger than the ones used for experiments on UCI datasets. This substantially increases the searching space of instance selection. One consequence in our experiments is that the *Discriminative* algorithm becomes very slow. Thus we were not able to produce comparison results for this algorithm. The proposed *Matrix* method was affected as well. However, we coped with this problem using a subsampling assisted method, where we first select a subset of 400 instances from the unlabeled set and then restrain our instance selection to this subset. This is equivalent to solving the matrix partition optimization in (12) with additional constraints on $M_b$, such that the columns of $M_b$ corresponding to instances outside of this subset of 400 instances are all set to 0. For the experiments, we chose the 400 instances as the ones with top entropy terms under the current classification model. The same subsampling was used for the method *Design* as well. Table 3 shows the comparison results on the three document categorization tasks, comparing *Matrix* to *svmD*, *Fisher*, *Design* and a baseline random selection, *Random*. These results show the proposed *Matrix* outperformed *svmD*, *Design* and *Random*. It tied with *Fisher* regarding overall measure, but had a slight *win* regarding pointwise measure.

These empirical results suggest that selecting unlabeled instances independent of the classification model using the proposed matrix partition method can achieve reasonable performance, which is better than a transductive experimental design method and comparable to the discriminative batch-mode active learning approaches. However, our approach can offer certain conveniences in some circumstances where one does not know the classification model to be employed for classification.

# 5 Conclusions

In this paper, we propose a novel batch-mode active learning approach that makes query selection decisions independent of the classification model employed. The proposed approach is based on a general *maximum mutual information* principle. It is formulated as a matrix partition optimization problem under a Gaussian process framework. To tackle the formulated combinatorial optimization problem, we developed an effective local optimization technique. Our empirical studies show the proposed flexible batch-mode active learning approach can achieve comparable or superior performance to discriminative batch-mode active learning methods that have been optimized on specific classifiers. A future extension for this work is to consider batch-mode active learning with structured data by exploiting different kernel functions.

# References

[1] S. Boyd and L. Vandenberghe. *Convex Optimization*. Cambridge University Press, 2004.

[2] K. Brinker. Incorporating diversity in active learning with support vector machines. In *Proceedings of International Conference on Machine learning*, 2003.

[3] T. Cover and J. Thomas. *Elements of Information Theory*. John Wiley & sons, 1991.

[4] C. Guestrin, A. Krause, and A. Singh. Near-optimal sensor placements in Gaussian processes. In *Proceedings of International Conference on Machine Learning*, 2005.

[5] Y. Guo and R. Greiner. Optimistic active learning using mutual information. In *Proceedings of International Joint Conference on Artificial Intelligence*, 2007.

[6] Y. Guo and D. Schuurmans. Discriminative batch mode active learning. In *Proceedings of Neural Information Processing Systems*, 2007.

[7] S. Hoi, R. Jin, and M. Lyu. Large-scale text categorization by batch mode active learning. In *Proceedings of the International World Wide Web Conference*, 2006.

[8] S. Hoi, R. Jin, J. Zhu, and M. Lyu. Batch mode active learning and its application to medical image classification. In *Proceedings of International Conference on Machine Learning*, 2006.

[9] S. Hoi, R. Jin, J. Zhu, and M. Lyu. Semi-supervised SVM batch mode active learning for image retrieval. In *Proceedings of IEEE Computer Society Conference on Computer Vision and Pattern Recognition*, 2008.

[10] A. Krause, C. Guestrin, A. Gupta, and J. Kleinberg. Near-optimal sensor placements: Maximizing information while minimizing communication cost. In *International Symposium on Information Processing in Sensor Networks*, 2006.

[11] C. Rasmussen and C. Williams. *Gaussian Processes for Machine Learning*. MIT Press, 2006.

[12] G. Schohn and D. Cohn. Less is more: Active learning with support vector machines. In *Proceedings of International Conference on Machine Learning*, 2000.

[13] B. Settles. Active learning literature survey. Computer Sciences Technical Report 1648, University of Wisconsin–Madison, 2009.

[14] Z. Xu, K. Yu, V. Tresp, X. Xu, and J. Wang. Representative sampling for text classification using support vector machines. In *European Conference on Information Retrieval*, 2003.

[15] K. Yu and J. Bi. Active learning via transductive experimental design. In *In Proceedings of the International Conference on Machine Learning*, 2006.

